# Q-MKL: Matrix-induced Regularization in Multi-Kernel Learning with Applications to Neuroimaging[*]

**Chris Hinrichs**[†‡]    **Vikas Singh**[†‡]    **Jiming Peng**[§]    **Sterling C. Johnson**[†‡]

[†]**University of Wisconsin**
Madison, WI

[§]**University of Illinois**
Urbana-Champaign, IL

[‡]**Geriatric Research Education & Clinical Center**
Wm. S. Middleton Memorial VA Hospital, Madison, WI

{ hinrichs@cs, vsingh@biostat, scj@medicine }.wisc.edu    pengj@illinois.edu

## Abstract

Multiple Kernel Learning (MKL) generalizes SVMs to the setting where one simultaneously trains a linear classifier and chooses an optimal combination of given base kernels. Model complexity is typically controlled using various norm regularizations on the base kernel mixing coefficients. Existing methods neither regularize nor exploit potentially useful information pertaining to how kernels in the input set 'interact'; that is, higher order kernel-pair relationships that can be easily obtained via unsupervised (similarity, geodesics), supervised (correlation in errors), or domain knowledge driven mechanisms (which features were used to construct the kernel?). We show that by substituting the norm penalty with an arbitrary quadratic function $\mathbf{Q} \succeq 0$, one can impose a desired covariance structure on mixing weights, and use this as an inductive bias when learning the concept. This formulation significantly generalizes the widely used 1- and 2-norm MKL objectives. We explore the model's utility via experiments on a challenging Neuroimaging problem, where the goal is to predict a subject's conversion to Alzheimer's Disease (AD) by exploiting aggregate information from many distinct imaging modalities. Here, our new model outperforms the state of the art ($p$-values $\ll 10^{-3}$). We briefly discuss ramifications in terms of learning bounds (Rademacher complexity).

## 1   Introduction

Kernel learning methods (such as Support Vector Machines) are conceptually simple, strongly rooted in statistical learning theory, and can often be formulated as a convex optimization problem. As a result, SVMs have come to dominate the landscape of supervised learning applications in bioinformatics, computer vision, neuroimaging, and many other domains. A standard SVM-based 'learning system' may be conveniently thought of as a composition of two modules [1, 2, 3, 4]: **(1)** Feature pre-processing, and **(2)** a core learning algorithm. The design of a kernel (feature pre-processing) may involve using different sets of extracted features, dimensionality reductions, or parameterizations of the kernel functions. Each of these alternatives produces a distinct kernel matrix. While much research has focused on efficient methods for the latter (i.e., support vector learning) step, specific choices of feature pre-processing are frequently a dominant factor in the system's overall performance as well, and may involve significant user effort. Multi-kernel learning [5, 6, 7] transfers a part of this burden from the user to the algorithm. Rather than selecting a single kernel, MKL offers the flexibility of specifying a large set of kernels corresponding to the many options (i.e., kernels) available, and additively combining them to construct an optimized, data-driven Reproducing

---

[*]Supported by NIH (R01AG040396), (R01AG021155); NSF (RI 1116584), (DMS 09-15240 ARRA), and (CMMI-1131690); Wisconsin Partnership Proposal; UW ADRC; UW ICTR (1UL1RR025011); AFOSR (FA9550-09-1-0098); and NLM (5T15LM007359). The authors would like to thank Maxwell Collins and Sangkyun Lee for many helpful discussions.

Kernel Hilbert Space (RKHS) – while *simultaneously* finding a max-margin classifier. MKL has turned out to be very successful in many applications: on several important Vision problems (such as image categorization), some of the best known results on community benchmarks come from MKL-type methods [8, 9]. In the context of our primary motivating application, the current state of the art in multi-modality neuroimaging-based Alzheimer's Disease (AD) prediction [10] is achieved by multi-kernel methods [3, 4], where each imaging modality spawns a kernel, or set of kernels.

In allowing the user to specify an arbitrary number of base kernels for combination MKL provides more expressive power, but this comes with the responsibility to regularize the kernel mixing coefficients so that the classifier generalizes well. While the importance of this regularization cannot be overstated, it is also a fact that commonly used $\ell_p$ norm regularizers operate on kernels separately, without explicitly acknowledging dependencies and interactions among them. To see how such dependencies can arise in practice, consider our neuroimaging learning problem of interest: the task of learning to predict the onset of AD. A set of base kernels $K_1, \ldots, K_M$ are derived from several different medical imaging modalities (MRI; PET), image processing methods (morphometric; anatomical modelling), and kernel functions (linear; RBF). Some features may be shared between kernels, or kernel functions may use similar parameters. As a result we expect the kernels' behaviors to exhibit some correlational, or other cluster structure according to how they were constructed. (See Fig. 2 (a) and related text, for a concrete discussion of these behaviors in our problem of interest.) We will denote this relationship as $\mathbf{Q} \in \mathbb{R}^{M \times M}$.

Ideally, the regularizer should reflect these dependencies encoded by $\mathbf{Q}$, as they can significantly impact the learning characteristics of a linearly combined kernel. Some extensions work at the level of group membership (e.g., [11]), but do not explicitly *quantify* these interactions. Instead, rather than penalizing covariances or inducing sparsity among groups of kernels, it may be beneficial to *reward* such covariances, so as to better reflect a latent cluster structure between kernels. In this paper, we show that a rich class of regularization schemes are possible under a new MKL formulation which regularizes on $\mathbf{Q}$ directly – the model allows one to exploit domain knowledge (as above) and statistical measures of interaction between kernels, employ estimated error covariances in ways that are not possible with $\ell_p$-norm regularization, or, encourage sparsity, group sparsity or non-sparsity as needed – all within a convex optimization framework. We call this form of multi-kernel learning, $\mathbf{Q}$-norm MKL or "$\mathbf{Q}$-MKL". This paper makes the following **contributions**: **(a)** presents our new $\mathbf{Q}$-MKL model which generalizes 1- (and 2-) norm MKL models, **(b)** provides a learning theoretic result showing that $\mathbf{Q}$-MKL can improve MKL's generalization error rate, **(c)** develops efficient optimization strategies (to be distributed with the *Shogun* toolbox), and **(d)** provides empirical results demonstrating statistically significant gains in accuracy on the important AD prediction problem.

**Background.** The development of MKL methods began with [5], which showed that the problem of learning the right kernel for an input problem instance could be formulated as a Semi-Definite Program (SDP). Subsequent papers have focused on designing more efficient optimization methods, which have enabled its applications to large-scale problem domains. To this end, the model in [5] was shown to be solvable as a Second Order Cone Program [12], a Semi-Infinite Linear Program [6], and via gradient descent methods in the dual and primal [7, 13]. More recently, efforts have focused on generalizing MKL to arbitrary $p$-norm regularizers where $p > 1$ [13, 14] while maintaining overall efficiency. In [14], the authors briefly mentioned that more general norms may be possible, but this issue was not further examined. A non-linear "hyperkernel" method was proposed [15] which implicitly maps the kernels themselves to an implicit RKHS, however this method is computationally very demanding, (it has $4^{th}$ order interactions among training examples). The authors of [16] proposed to first select the sub-kernel weights by minimizing an objective function derived from Normalized Cuts, and subsequently train an SVM on the combined kernel. In [17, 2], a method was proposed for selecting an optimal finite combination from an infinite parameter space of kernels. Contemporary to these results, [18] showed that if a large number of kernels had a desirable shared structure (e.g., followed directed acyclic dependencies), extensions of MKL could still be applied. Recently in [8], a set of base classifiers were first trained using each kernel and were then boosted to produce a strong multi-class classifier. At this time, MKL methods [8, 9] provide some of the best known accuracy on image categorization datasets such as Caltech101/256 (see `www.robots.ox.ac.uk/~vgg/software/MKL/`). Next, we describe in detail the motivation and theoretical properties of $\mathbf{Q}$-MKL .

## 2 From MKL to Q-MKL

**MKL Models.** Adding kernels corresponds to taking a direct sum of Reproducing Kernel Hilbert spaces (RKHS), and scaling a kernel by a constant $c$ scales the *axes* of it's RKHS by $\sqrt{c}$. In the MKL setting, the SVM margin regularizer $\frac{1}{2}\|\mathbf{w}\|^2$ becomes a weighted sum $\frac{1}{2}\sum_{m=1}^{M}\frac{\|\mathbf{w}_m\|_{\mathcal{H}_m}^2}{\beta_m}$ over contributions from RKHS's $\mathcal{H}_1, \ldots, \mathcal{H}_M$, where the vector of mixing coefficients $\beta$ scales each respective RKHS [14]. A norm penalty on $\beta$ ensures that the units in which the margin is measured are meaningful (provided the base kernels are normalized). The MKL primal problem is given as

$$\min_{\mathbf{w}, b, \beta \geq 0, \xi \geq 0} \frac{1}{2}\sum_{m}^{M}\frac{\|\mathbf{w}_m\|_{\mathcal{H}_m}^2}{\beta_m} + C\sum_{i}^{n}\xi_i + \|\beta\|_p^2 \quad \text{s.t. } y_i\left(\sum_{m}^{M}\langle\mathbf{w}_m, \phi_m(\mathbf{x}_i)\rangle_{\mathcal{H}_m} + b\right) \geq 1 - \xi_i, \quad (1)$$

where $\phi_m(\mathbf{x})$ is the (potentially unknown) transformation from the original data space to the $m^{\text{th}}$ RKHS $\mathcal{H}_m$. As in SVMs, we turn to the dual problem to see the role of kernels:

$$\max_{0 \leq \alpha \leq C} \alpha^T\mathbf{1} - \frac{1}{2}\|\mathbb{G}\|_q, \quad \mathbb{G} \in \mathbb{R}^M; \mathbb{G}_m = (\alpha \circ \mathbf{y})^T K_m(\alpha \circ \mathbf{y}), \quad (2)$$

where $\circ$ denotes element-wise multiplication, and the dual $q$-norm follows the identity $\frac{1}{p} + \frac{1}{q} = 1$. Note that the primal norm penalty $\|\beta\|_p^2$ becomes a dual-norm on the vector $\mathbb{G}$. At optimality, $\mathbf{w}_m = \beta_m(\alpha \circ \mathbf{y})^T\phi_m(X)$, and so the term $\mathbb{G}_m = (\alpha \circ \mathbf{y})^T K_m(\alpha \circ \mathbf{y}) = \frac{\|\mathbf{w}_m\|_{\mathcal{H}_m}^2}{\beta_m^2}$ is the vector of *scaled* classifier norms. This shows that the dual norm is tied to how MKL measures the margin in each RKHS.

**The Q-MKL model.** The key characteristic of **Q**-MKL is that the standard (squared) $\ell_p$-norm penalty on $\beta$, along with the corresponding dual-norm penalty in (2), is substituted with a more general class of quadratic penalty functions, expressed as $\beta^T\mathbf{Q}\beta = \|\beta\|_{\mathbf{Q}}^2$. $\|\beta\|_{\mathbf{Q}} = \sqrt{\beta^T\mathbf{Q}\beta}$ is a Mahalanobis (matrix-induced) norm so long as $\mathbf{Q} \succeq 0$. In this framework, the burden of choosing a kernel is deferred to a choice of **Q**-function. This approach gives the algorithm greater flexibility while controlling model complexity, as we will discuss shortly. The model we optimize is,

$$\min_{w, b, \beta \geq 0, \xi \geq 0} \frac{1}{2}\sum_{m}^{M}\frac{\|\mathbf{w}_m\|_{\mathcal{H}_m}^2}{\beta_m} + C\sum_{i}^{n}\xi_i + \beta^T\mathbf{Q}\beta \quad \textbf{s.t. } y_i\left(\sum_{m}^{M}\langle\mathbf{w}_m, \phi_m(\mathbf{x}_i)\rangle_{\mathcal{H}_m} + b\right) \geq 1 - \xi_i, \quad (3)$$

where the last objective term provides a bias relative to $\beta^T\mathbf{Q}\beta$. The dual problem becomes $\max_\alpha \alpha^T\mathbf{1} - \frac{1}{2}\sqrt{\mathbb{G}^T\mathbf{Q}^{-1}\mathbb{G}}$. It is easy to see that if $\mathbf{Q} = \mathbf{1}^{M \times M}$, we obtain the $p = 1$ form of (1), i.e., 1-norm MKL, as a special case because $\beta^T\mathbf{1}^{M \times M}\beta = \|\beta\|_1^2$. On the other hand, setting $\mathbf{Q}$ to $\mathbb{I}^{M \times M}$ (identity), reduces to 2-norm MKL.

## 3 The case for Q-MKL

Extending the MKL regularizer to arbitrary quadratics $\mathbf{Q} \succeq 0$ significantly expands the richness of the MKL framework; yet we can show that for reasonable choices of $\mathbf{Q}$, this actually *decreases* MKL's learning-theoretic complexity. Joachims *et al.* [19] derived a theoretical generalization error bound on kernel combinations which depends on the degree of redundancy between support vectors in SVMs trained on base kernels individually. Using this type of correlational structure, we can derive a $\mathbf{Q}$ function between kernels to automatically select a combination of kernels which will maximize this bound. This type of $\mathbf{Q}$ function can be shown to have lower Rademacher complexity, (see below,) while simultaneously decreasing the error bound from [19], which does not directly depend on Rademacher complexity.

### 3.1 Virtual Kernels, Rademacher Complexity and Renyi Entropy

If we decompose $\mathbf{Q}$ into its component eigen-vectors, we can see that each eigen-vector defines a linear combination of kernels. This observation allows us to analyze $\mathbf{Q}$-MKL in terms of these objects, which we will refer to as **Virtual Kernels**. We first show that as $\mathbf{Q}^{-1}$'s eigen-values decay, so do the traces of the virtual kernels. Assuming $\mathbf{Q}^{-1}$ has a bounded, non-uniform spectrum, this property can then be used to analyze, (and bound), $\mathbf{Q}$-MKL's Rademacher complexity, which has been shown to depend on the traces of the base kernels. We then offer a few observations on how $\mathbf{Q}^{-1}$'s Renyi entropy [20] relates to these learning theoretic bounds.

**Virtual Kernels.** In the following, assume that $\mathbf{Q} \succ 0$, and has eigen-decomposition $\mathbf{Q} = V\Lambda V$, with $V = \{v_1, \cdots, v_M\}$. First, observe that because $\mathbf{Q}$'s eigen-vectors provide an orthonormal basis of $\mathbb{R}^M$, $\beta \in \mathbb{R}^M$ can be expressed as a linear combination in this basis with $\gamma$ as its coefficients: $\beta = \sum_i \gamma_i v_i = V\gamma$. Substituting in $\beta^T \mathbf{Q}\beta$ we have

$$\beta^T \mathbf{Q}\beta = (\gamma^T V^T) V\Lambda V^T (V\gamma) = \gamma^T (V^T V)\Lambda(V^T V)\gamma = \gamma^T \Lambda \gamma = \sum_i \gamma_i^2 \lambda_i \tag{4}$$

This simple observation offers an alternate view of what $\mathbf{Q}$-MKL is actually optimizing. Each eigen-vector $v_i$ of $\mathbf{Q}$ can be used to define a linear combination of kernels, which we will refer to as virtual kernel $\widetilde{\mathbb{K}}_i = \sum_m v_i(m) K_m$. Note that if $\widetilde{\mathbb{K}}_i \succeq 0$, $\forall i$, then they each define an independent RKHS. This can be ensured by choosing $\mathbf{Q}$ in a specific way, if desired. This leads to the following result:

**Lemma 1.** *If $\widetilde{\mathbb{K}}_i \succeq 0, \forall i$, then $\mathbf{Q}$-MKL is equivalent to 2-norm MKL using virtual kernels instead of base kernels.*

*Proof.* Let $\mu_i = \gamma_i \sqrt{\lambda_i}$. Then $\beta^T \mathbf{Q}\beta = \|\mu\|_2^2$, (eq. 4) and $K^* = \sum_m \beta_m K_m = \sum_m^M \sum_i^M \gamma_i v_i(m) K_m = \sum_i^M \mu_i \lambda^{-\frac{1}{2}} \sum_m^M v_i(m) K_m = \sum_i^M \mu_i \widetilde{\mathbb{K}}_i$, where $\widetilde{\mathbb{K}}_i = \lambda^{-\frac{1}{2}} \sum_m^M v_i(m) K_m$ is the $i^{th}$ virtual kernel. The learned kernel $K^*$ is a weighted combination of virtual kernels, and the coefficients are regularized under a squared 2-norm. $\square$

**Rademacher Complexity in MKL.** With this result in hand, we can now evaluate the Rademacher complexity of $\mathbf{Q}$-MKL by using a recent result for $p$-norm MKL. We first state a theorem from [21], which relates the Rademacher complexity of MKL to the traces of its base kernels.

**Theorem 1.** *([21]) The empirical Rademacher complexity on a sample set $S$ of size $n$, with $M$ base kernels is given as follows (with $\eta_0 = \frac{23}{22}$),*

$$\mathfrak{R}_S(\mathcal{H}_M{}^p) \leq \frac{\sqrt{\eta_0 q \|\mathbf{u}\|_q}}{n} \tag{5}$$

*where $\mathbf{u} = [\mathrm{Tr}(K_1), \cdots, \mathrm{Tr}(K_M)]^T$ and $\frac{1}{p} + \frac{1}{q} = 1$.*

The bound in (5) shows that the Rademacher complexity $\mathfrak{R}_S(\cdot)$ depends on $\|\mathbf{u}\|_q$, a norm on the base kernels' traces. Assuming they are normalized to have unit trace, the bound for $p = q = 2$-norm MKL is governed by $\|\mathbf{u}\|_2 = \sqrt{M}$. However, in $\mathbf{Q}$-MKL the virtual kernels traces are not equal, and are in fact given by $\mathrm{Tr}(\widetilde{\mathbb{K}}_i) = \frac{\mathbf{1}^T v_i}{\sqrt{\lambda_i}}$. With this expression for the traces of the virtual kernels, we can now prove that the bound given in (5) is strictly decreased as long as the eigen-values $\psi_i$ of $\mathbf{Q}^{-1}$ are in the range $(0, 1]$. (Adding 1 to the diagonal of $\mathbf{Q}$ is sufficient to guarantee this.)

**Theorem 2.** *If $\mathbf{Q}^{-1} \neq \mathbb{I}^{M \times M}$ and $\widetilde{\mathbb{K}}_i \succeq 0 \; \forall i$ then the bound on Rademacher complexity given in (5) is strictly lower for $\mathbf{Q}$-MKL than for 2-norm MKL.*

*Proof.* By Lemma 1, we have that the bound in (5) will decrease if $\|\mathbf{u}\|_2$, the norm on the virtual kernel traces, decreases. As shown above, the virtual kernel traces are given as $\mathrm{Tr}(\widetilde{\mathbb{K}}_i) = \sqrt{\psi_i} \mathbf{1}^T v_i$, meaning that $\|\mathbf{u}\|_2^2 = \sum_i^N \psi_i (\mathbf{1}^T v_i)^2 = \sum_i^N \psi_i \mathbf{1}^T v_i v_i^T \mathbf{1} = \mathbf{1}^T \mathbf{Q}^{-1} \mathbf{1}$. Clearly, this sum is maximal for $\psi_i = 1$, $\forall i$, which is true if and only if $\mathbf{Q}^{-1} = \mathbb{I}^{M \times M}$. This means that when $\mathbf{Q} \neq \mathbb{I}^{M \times M}$, then the bound in (5) is strictly decreased. $\square$

Note that requiring the virtual kernels to be p.s.d., while achievable (see supplements,) is somewhat restrictive. In practice, such a $\mathbf{Q}$ matrix may not differ substantially from $\mathbb{I}^{N \times N}$. We therefore provide the following result which frees us from this restriction, and has more practical significance.

**Theorem 3.** $\mathbf{Q}$-*MKL is equivalent to the following model:*

$$\min_{\mathbf{w}, b, \mu, \xi \geq 0} \quad \frac{1}{2} \sum_m^M \frac{\|\mathbf{w}_m\|_{\mathcal{V}_m}^2}{\mu_m} + C \sum_i^n \xi_i + \|\mu\|_2^2 \tag{6}$$

$$s.t. \quad y_i \left( \sum_m^M \langle \mathbf{w}_m, \phi_m(\mathbf{x}_i) \rangle_{\mathcal{V}_m} + b \right) \geq 1 - \xi_i, \quad \mathbf{Q}^{-\frac{1}{2}} \mu \geq 0,$$

*where $\phi_m()$ is the feature transform mapping data space to the $m^{th}$ virtual kernel, denoted as $\mathcal{V}_m$.*

While the virtual kernels themselves may be indefinite, recall that $\mu = \mathbf{Q}^{\frac{1}{2}}\beta$, and so the constraint $\mathbf{Q}^{-\frac{1}{2}}\mu \geq 0$ is equivalent to $\beta \geq 0$, guaranteeing that the combined kernel will be p.s.d. This formulation is slightly different than the 2-norm MKL formulation, however it does not alter the theoretical guarantee of [21], providing a stronger result.

**Renyi Entropy.** Renyi entropy [20] significantly generalizes the usual notion of Shannon entropy [22, 23, 24], has applications in Statistics and many other fields, and has recently been proposed as an alternative to PCA [22]. Thm. 2 points to an intuitive explanation of where the benefit from a $\mathbf{Q}$ regularizer comes from as well, if we analyze the Renyi entropy of the distribution on kernels defined by $\mathbf{Q}^{-1}$, if we treat $\mathbf{Q}^{-1}$ as a kernel density estimator. The quadratic Renyi entropy of a probability measure is given as,

$$H(p) = -\log \int p^2(\mathbf{x})d\mathbf{x}.$$

Now, if we use a kernel function (i.e., $\mathbf{Q}^{-1}$), and a finite sample (i.e., base kernels), as a kernel density estimator, (*cf.* [15],) then with some normalization we can derive an estimate of the underlying probability $\hat{p}$, which is a distribution over base kernels. We can then interpret its Renyi entropy as a complexity measure on the space of combined kernels. Eq. (5.2) in [23] relates the virtual kernel traces to the Renyi entropy estimator of $\mathbf{Q}^{-1}$ as $\int \hat{p}^2(\mathbf{x})d\mathbf{x} = \frac{1}{N^2}\mathbf{1}^T\mathbf{Q}^{-1}\mathbf{1}$,[1] which leads to a nice connection to Thm. 2. This view informs us that setting $\mathbf{Q}^{-1} = \mathbb{I}^{M \times M}$, (i.e., 2-norm MKL), has *maximal* Renyi entropy because it is maximally uninformative; adding structure to $\mathbf{Q}^{-1}$ concentrates $\hat{p}$, reducing both its Renyi entropy, *and* Rademacher complexity together.

This series of results suggests an entirely new approach to analyzing the Rademacher complexity of MKL methods. The proof of Thm. 2 relies on decreasing a norm on the virtual kernel traces, which we now see directly relates to the Renyi entropy of $\mathbf{Q}^{-1}$, as well as with decreasing the Rademacher complexity of the search space of combined kernels. It is even possible that by directly analyzing Renyi entropy in a multi-kernel setting, this conjecture may be useful in deriving analogous bounds in, *e.g.*, Indefinite Kernel Learning [25], because the virtual kernels are indefinite in general.

### 3.2 Special Cases: $\mathbf{Q}$-SVM and relative margin

Before describing our optimization strategy, we discuss several variations on the $\mathbf{Q}$-MKL model.

**$\mathbf{Q}$-SVM.** An interesting special case of $\mathbf{Q}$-MKL is $\mathbf{Q}$-SVM, which generalizes several recent, (but independently developed,) models in the literature [26, 27]. If the base kernels are rank-1, (i.e., singleton features,) then each coefficient $\beta_m$ effectively becomes a feature weight, and a 2-norm penalty on $\beta$ is a penalty on weights. $\mathbf{Q}$-MKL therefore reduces to a form of SVM in which $\|\mathbf{w}\|^2$ becomes $\mathbf{w}^T\mathbf{Q}\mathbf{w}$. Thus, in such cases we can reduce the $\mathbf{Q}$-MKL model to a simple QP, which we call $\mathbf{Q}$-SVM . Please refer to the supplements for details, and some experimental results.

**Relative Margin.** Several interesting extensions to the SVM and MKL frameworks have been proposed which focus on the *relative* margin methods [28, 29] which maximize the margin relative to the spread of the data. In particular $\mathbf{Q}$-MKL can be easily modified to incorporate the Relative Margin Machine (RMM) model [28] by replacing Module 1 as in (7) with the RMM objective. Our alternating optimization approach, (described next,) is not affected by this addition; however, the additional constraints would mean that SMO-based strategies would not be applicable.

## 4 Optimization

We now present the core engine to solve (3). Most MKL implementations make use of an alternating minimization strategy which first minimizes the objective in terms of the SVM parameters, and then with respect to the sub-kernel weights $\beta$. Since the MKL problem is convex, this method leads to global convergence [7, 14] and minor modifications to standard SVM implementations are sufficient. $\mathbf{Q}$-MKL generalizes $\|\beta\|_p^2$ to arbitrary convex quadratic functions, while the feasible set is the same as for MKL. This directly gives that the $\mathbf{Q}$-MKL model in (3) is convex. We will broadly follow this strategy, but as will become clear shortly, interaction between sub-kernel weights makes the optimization of $\beta$ more involved (than [6, 14]), and requires alternative solution mechanisms. We may consider this process as a composition of two modules: one which solves for SVM dual parameters ($\alpha$) with fixed $\beta$, and the other for solving for $\beta$ with fixed $\alpha$:

| **(Module 1)** | **(Module 2)** |
|---|---|
| $$\max_{0 \le \alpha \le C} \alpha^T \mathbf{1} - \alpha^T Y K Y \alpha \quad \text{s.t.} \alpha^T y = 0 \quad (7)$$ | $$\min_{\beta \ge 0} \sum_m \frac{\|\mathbf{w}_m\|^2}{\beta_m} \quad \text{s.t.} \beta^T \mathbf{Q} \beta \le 1 \quad (8)$$ |

Using a result from [14] we can replace the $\beta^T \mathbf{Q} \beta$ objective term with a quadratic constraint, which gives the problem in (8). Notice that (8) has a sum of ratios with optimization variables in the denominator, while the constraint is quadratic – this means that standard convex optimization toolkits may not be able to solve this problem without significant reformulation from its canonical form in (8). Our approach is to search for a stationary point by representing the gradient as a non-linear system. Writing the gradient in terms of the Lagrange multiplier $\delta$, and setting it equal to 0 gives:

$$\frac{\|\mathbf{w}_m\|^2_{\mathcal{H}_m}}{\beta_m^2} - \delta(\mathbf{Q}\beta)_m = 0, \quad \forall m \in \{1, \cdots, M\}. \tag{9}$$

We now seek to eliminate $\delta$ so that the non-linear system will be limited to quadratic terms in $\beta$, allowing us to use a non-linear system solver. Let $\mathbf{W} = \text{Diag}(\|\mathbf{w}_1\|^2_{\mathcal{H}_1}, \ldots, \|\mathbf{w}_M\|^2_{\mathcal{H}_M})$, and $\beta^{-2} = (\beta_1^{-2}, \ldots, \beta_M^{-2})$. We can then write $\mathbf{W}\beta^{-2} = \delta(\mathbf{Q}\beta)$. Now, solving for $\beta$ (on the right hand side) gives

$$\beta = \frac{1}{\delta} \mathbf{Q}^{-1} \mathbf{W} \beta^{-2} \tag{10}$$

Because $\mathbf{Q} \succ 0$, and $\beta \ge 0$, at optimality the constraint $\beta^T \mathbf{Q} \beta \le 1$ must be active. So, we can plug in the above identity to solve for $\delta$,

$$1 = \left(\frac{1}{\delta} \mathbf{Q}^{-1} \mathbf{W} \beta^{-2}\right)^T \mathbf{Q} \left(\frac{1}{\delta} \mathbf{Q}^{-1} \mathbf{W} \beta^{-2}\right)$$
$$\delta = \sqrt{(\mathbf{W}\beta^{-2})^T \mathbf{Q}^{-1} (\mathbf{W}\beta^{-2})} = \|\mathbf{W}\beta^{-2}\|_{\mathbf{Q}^{-1}}, \tag{11}$$

which shows that $\delta$ effectively normalizes $\mathbf{W}\beta^{-2}$ according to $\mathbf{Q}^{-1}$. We can now solve (10) in terms of $\beta$ using a nonlinear root finder, such as the GNU Scientific Library; in practice this method is quite efficient, typically requiring 10 to 20 outer iterations. Putting these parts together, we can propose following algorithm for optimizing $\mathbf{Q}$-MKL:

---

**Algorithm 1. Q-*MKL***

*Input: Kernels* $\{K_1, \cdots, K_M\}$*;* $\mathbf{Q} \succeq 0 \in \mathbb{R}^{M \times M}$*; labels* $y \in \{\pm 1\}^N$*.*
*Outputs:* $\alpha$*, b,* $\beta$
$\beta^{(0)} = \frac{1}{M}$*;* $t = 0$ *(iterations)*
**while** *not optimal* **do**
    $K^{(t)} \leftarrow \sum_m \beta_m^{(t)} K_m$
    $\alpha^{(t)}, b^{(t)} \leftarrow \text{SVM}(K^{(t)}, C, y)$ **(Module 1***, (7))*
    $W_{mm} = \alpha^{(t)T} K_m^{(t)} \alpha^{(t)} (\beta_m^{(t)})^2$
    $\beta^{(t+1)} \leftarrow \arg\min(\text{Problem}(8))$ **(Module 2***, (8))*
    $t = t + 1$
**end while**

---

### 4.1 Convergence

We can show that our model can be solved optimally by noting that Module 2 can be precisely optimized at each step. If Module 2 *cannot* be solved precisely, then Algorithm 1 may not converge. The following result assures us that indeed Module 2 can be solved precisely by reducing it to a convex Semi-Definite Program (SDP).

**Theorem 4.** *The solution to Problem (8) is the same as the solution to the following SDP:*

$$\min_{\nu \ge 0, \beta \ge 0, Z \in \mathbb{R}^{M \times M}} w^T \nu \tag{12}$$

$$\text{subject to} \quad \begin{bmatrix} \nu_m & 1 \\ 1 & \beta_m \end{bmatrix} \succeq 0, \ \forall m \qquad \begin{bmatrix} 1 & \beta^T \\ \beta & Z \end{bmatrix} \succeq 0, \qquad \text{Tr}(\mathbf{Q}Z) \le 1. \tag{13}$$

*Proof.* The first PSD constraint (13) requires that $\nu_m = \beta_m^{-1}$, meaning that objective (12) is the same as that of Problem (8). From the second we have $Z = \beta\beta^T$, and so $\text{Tr}(\mathbf{Q}Z) = \beta^T \mathbf{Q}\beta$; therefore the feasible sets are equivalent. $\square$

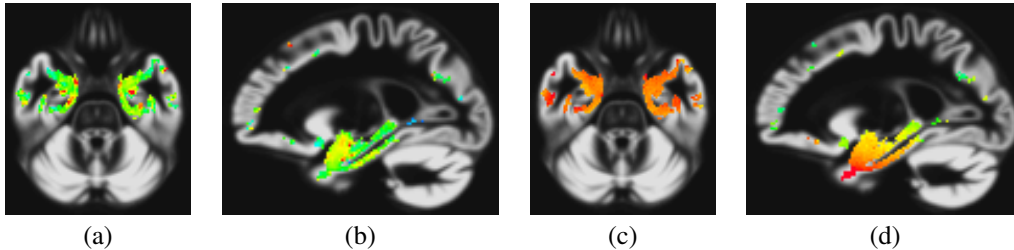

Figure 1: Comparison of spatial smoothness of weights chosen by **Q**-SVM and SVM with gray matter (GM) density maps. Left (a-b): weights given by a standard SVM; Right (c-d): weights given by **Q**-SVM .

The last PSD constraint is only necessary to ensure that $\beta^T \mathbf{Q} \beta \leq 1$, and can be replaced with that quadratic constraint. Doing so yields a Second-Order Cone Program (SOCP) which is also amenable to standard solvers. Note that it is not necessary to solve for $\beta$ as an SDP, though it may nevertheless be an effective solution mechanism, depending on the size and characteristics of the problem.

## 5   Experiments

We performed extensive experiments to validate **Q**-MKL, examine the effect it has on $\beta$, and to assess its advantages in the context of our motivating neuroimaging application. In these main experiments, we demonstrate how domain knowledge can be adapted to improve the algorithm's performance. Our focus on a practical application is intended as a demonstration of how domain knowledge can be seamlessly incorporated into a learning model, giving significant gains in accuracy. We also performed experiments on the UCI repositories, which are described in detail in the supplements. Briefly, in these experiments **Q**-MKL performed as well as, or better than, 1- and 2-norm MKL on most datasets, showing that even in the absence of significant domain knowledge, **Q**-MKL can still perform about as well as existing MKL methods.

**Image preprocessing.** In out main experiments we used brain scans of AD patients and Cognitively Normal healthy controls (CN) from the Alzheimer's Disease Neuroimaging Initiative (ADNI) [30] in a set of cross-validation experiments. ADNI is a landmark study sponsored by the NIH, major pharmaceuticals and others to determine the extent to which multi-modal brain imaging can help predict on-set, and monitor progression of, AD. To this end, MKL-type methods have already defined the state of the art for this application [3, 4]. For our experiments, 48 AD subjects and 66 controls were chosen who had both $T_1$-weighted MR scans and Fluoro-Deoxy-Glucose PET (FDG-PET) scans at two time-points two years apart. Standard diffeomorphic methods, known generally as Voxel-Based Morphometry (VBM), (see SPM, `www.fil.ion.ucl.ac.uk/spm/`) were used to register scans to a common template and calculate Gray Matter (GM) densities at each voxel in the MR scans. We also used Tensor-Based Morphometry (TBM) to calculate maps of longitudinal voxel-wise expansion or contraction over a two year period. Feature selection was performed separately in each set of images by sorting voxels by $t$-statistic (calculated using training data), and choosing the highest 2000, 5000, 10000,...,250000 voxels in 8 stages. We used linear, quadratic, and Gaussian kernels: a total of 24 kernels per set, (GM maps, TBM maps, baseline FDG-PET, FDG-PET at 2-year follow up) for a total of 96 kernels. For **Q**-matrix we used the Laplacian of covariance between single-kernel $\alpha$ parameters, (recall the motivation from [19] in Section 3,) plus a block-diagonal representing clusters of kernels derived from the same imaging modalities.

### 5.1   Spatial SVM

Before describing out main experiments, we first return to the **Q**-SVM model briefly mentioned in 3.2. To demonstrate that **Q**-regularizers indeed influence the learned classifier, we performed classification experiments with the Laplacian of the inverse distance between voxels as a **Q** matrix, and voxel-wise GM density (VBM) as features. Using 10-fold cross-validation with 10 realizations, **Q**-SVM 's accuracy was 0.819, compared to the regular SVM's accuracy of 0.792. These accuracies are significantly different at the $\alpha = 0.0005$ level under a paired $t$-test. In Fig. 1 we show a comparison of weights trained by a regular SVM (a–b), and those trained by a spatially regularized SVM, (c–d). Note the greater spatial smoothness, and lower incidence of isolated "pockets".

## 5.2 Multi-modality Alzheimer's disease (AD) prediction

Next, we performed multi-modality AD prediction experiments using all 96 kernels across two modalities: MR provides structural information, while FDG-PET assesses hypo-metabolism. Further, we may use several image processing pipelines. Due to the inherent similarities in how the various kernels are derived, there are clear cluster structures / behaviors among the kernels, which we would like to exploit using **Q**-MKL. We used 10-fold cross-validation with 30 realizations, for a total of 300 folds. Accuracy, sensitivity and specificity were averaged over all folds. For comparison we also examined 1-, 1.5-, and 2-norm MKL. As MKL methods

| Regularizer | Acc. | Sens. | Spec. |
|---|---|---|---|
| $\|\beta\|_1$-MKL | 0.864 | 0.771 | 0.931 |
| $\|\beta\|_{1.5}$-MKL | 0.875 | **0.790** | 0.936 |
| $\|\beta\|_2$-MKL | 0.875 | **0.789** | 0.938 |
| $\text{Cov}_\alpha$ | **0.884** | 0.780 | 0.942 |
| Lap.($\text{Cov}_\alpha$) | **0.884** | 0.785 | **0.955** |
| Lap.($\text{Cov}_\alpha$) + diag | **0.888** | 0.786 | **0.956** |

Table 1: Comparison of **Q**-MKL & MKL. Bold numerals indicate methods not differing from the best at the 0.01 level using a paired $t$-test. Lap. = "Laplacian"; diag = "Block-diagonal".

have emerged as the state of the art in this domain [3, 4], and have performed favorably in extensive evaluations against various baselines such as single-kernel methods, and naïve combinations, we therefore focus our analysis on comparison with existing MKL methods. Results are shown in Table 1. **Q**-MKL had the highest performance overall, reducing the error rate from 12.5% to 11.2%. (Significant at the $\alpha = 0.001$ level.) Note that the *in vivo* diagnostic error rate for AD is believed to be near 8–10%, meaning that this improvement is quite significant. The primary benefit of current sparse MKL methods is that they effectively *filter out* uninformative or noisy kernels, however, the kernels used in these experiments are all derived from clinically relevant neuroimaging data, and are thus highly reliable. **Q**-MKL's performance suggests that it boosts the overall accuracy.

**Virtual kernel analysis.** We next turn to an analysis of the covariance structures found in the data empirically as a concrete demonstration of the type of patterns towards which the **Q**-MKL regularizer biases $\beta$. Recall that **Q**'s eigen-vectors can show which patterns are encouraged or deterred, in proportion to their eigen-values. In Fig. 2, we compare the **Q** matrix used in the ADNI experiments, based on the correlations of single-kernel $\alpha$ parameters (a), the 3 least eigenvectors of its graph Laplacian (b–d), and the $\beta$ vector optimized by **Q**-MKL . In (a), we can see that while the VBM (first block of 24 kernels) and TBM (second block of kernels) are highly correlated, they appear to be fairly uncorrelated to one another. The FDG-PET kernels (last 48 kernels) are much more strongly interrelated. Interestingly, the first eigenvector is almost entirely devoted to two large blocks of kernels – those which come from MRI data, and those which come from FDG-PET data. The positive elements in the off-diagonal encourage sparsity within these two super-blocks of kernels. Somewhat to the contrary, the next two eigenvecors have negative weights in the region between TBM and VBM kernels, encouraging non-sparsity between these two blocks. In (e) we see that the optimized $\beta$ discards most TBM kernels, (but not all,) putting the strongest weight on a few VBM kernels, and keeps a wider distribution of the FDG-PET kernels.

**Conclusion.** MKL is an elegant method for aggregating multiple data views, and is being extensively adopted for a variety of problems in machine learning, computer vision, and neuroimaging. **Q**-MKL extends this framework to exploit higher order interactions between kernels using supervised, unsupervised, or domain-knowledge driven measures. This flexibility can impart greater control over how the model utilizes cluster structure among kernels, and effectively encourage cancellation of errors wherever possible. We have presented a convex optimization model to efficiently solve the resultant model, and shown experiments on a challenging problem of identifying AD based on multi modal brain imaging data (obtaining statistically significant improvements). Our implementation will be made available within the Shogun toolbox (www.shogun-toolbox.org).

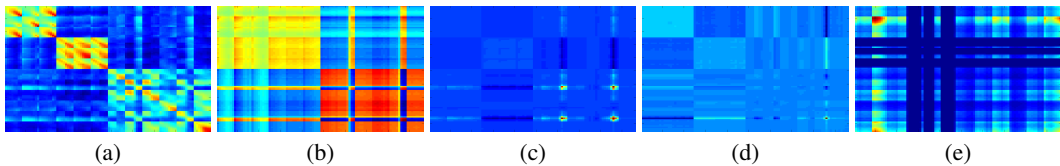

|     |     |     |     |     |
|-----|-----|-----|-----|-----|
| (a) | (b) | (c) | (d) | (e) |

Figure 2: Cov. **Q** used in AD experiments (a); three least graph Laplacian eigen-vectors (b-d); outer product of optimized $\beta$ (e). Note the block structure in (a–d) relating to the imaging modalities and kernel functions.

## Footnotes

[1] Note that this involves a Gaussian assumption, but [24] provides extensions to non-Gauss kernels.

# References

[1] I. Guyon and A. Elisseeff. An introduction to variable and feature selection. *JMLR*, 3:1157–1182, 2003.

[2] P. V. Gehler and S. Nowozin. Let the kernel figure it out; principled learning of pre-processing for kernel classifiers. *CVPR*, 2009.

[3] C. Hinrichs, V. Singh, G. Xu, and S.C. Johnson. Predictive markers for AD in a multi-modality framework: An analysis of MCI progression in the ADNI population. *Neuroimage*, 55(2):574–589, 2011.

[4] D. Zhang, Y. Wang, L. Zhou, H. Yuan, and D. Shen. Multimodal Classification of Alzheimer's Disease and Mild Cognitive Impairment. *NeuroImage*, 55(3):856–867, 2011.

[5] G. R. G. Lanckriet, N. Cristianini, P. Bartlett, L. El Ghaoui, and M. Jordan. Learning the kernel matrix with semidefinite programming. *JMLR*, 5:27–72, 2004.

[6] S. Sonnenburg, G. Rätsch, C. Schäfer, and B. Schölkopf. Large scale multiple kernel learning. *JMLR*, 7:1531–1565, 2006.

[7] A. Rakotomamonjy, F. Bach, S. Canu, and Y. Grandvalet. SimpleMKL. *JMLR*, 9:2491–2521, 2008.

[8] P. V. Gehler and S. Nowozin. On feature combination for multiclass object classification. In *ICCV*, 2009.

[9] J. Yang, Y. Li, Y. Tian, L. Duan, and W. Gao. Group-sensitive multiple kernel learning for object categorization. In *ICCV*, 2009.

[10] P. Vemuri, J.L. Gunter, M. L. Senjem, J. L. Whitwell, K. Kantarci, D. S. Knopman, et al. Alzheimer's disease diagnosis in individual subjects using structural MR images: validation studies. *Neuroimage*, 39(3):1186–1197, 2008.

[11] M. Szafranski, Y. Grandvalet, and A. Rakotomamonjy. Composite kernel learning. *Machine learning*, 79(1):73–103, 2010.

[12] F. R. Bach, G. Lanckriet, and M. I. Jordan. Multiple kernel learning, conic duality, and the SMO algorithm. In *ICML*, 2004.

[13] F. Orabona, L. Jie, and B. Caputo. Online-Batch Strongly Convex Multi Kernel Learning. In *CVPR*, 2010.

[14] M. Kloft, U. Brefeld, S. Sonnenburg, and A. Zien. $\ell_p$-Norm Multiple Kernel Learning. *JMLR*, 12:953–997, 2011.

[15] C.S. Ong, A. Smola, and B. Williamson. Learning the kernel with hyperkernels. *JMLR*, 6:1045–1071, 2005.

[16] L. Mukherjee, V. Singh, J. Peng, and C. Hinrichs. Learning Kernels for variants of Normalized Cuts: Convex Relaxations and Applications. *CVPR*, 2010.

[17] P. V. Gehler and S. Nowozin. Infinite kernel learning. Technical Report 178, Max-Planck Institute for Biological Cybernetics, 10 2008.

[18] F. R. Bach. Exploring large feature spaces with hierarchical multiple kernel learning. In *NIPS*, 2008.

[19] T. Joachims, N. Cristianini, and J. Shawe-Taylor. Composite kernels for hypertext categorisation. In *ICML*, 2001.

[20] A. Renyi. On measures of entropy and information. In *Fourth Berkeley Symposium on Mathematical Statistics and Probability*, pages 547–561, 1961.

[21] C. Cortes, M. Mohri, and A. Rostamizadeh. Generalization bounds for learning kernels. In *ICML*, 2010.

[22] R. Jenssen. Kernel entropy component analysis. *IEEE Trans. PAMI*, pages 847–860, 2009.

[23] M. Girolami. Orthogonal series density estimation and the kernel eigenvalue problem. *Neural Computation*, 14(3):669–688, 2002.

[24] D. Erdogmus and J.C. Principe. Generalized information potential criterion for adaptive system training. *IEEE Trans. Neural Networks*, 13(5):1035–1044, 2002.

[25] M. Kowalski, M. Szafranski, and L. Ralaivola. Multiple indefinite kernel learning with mixed norm regularization. In *ICML*, 2009.

[26] S. Bergsma, D. Lin, and D. Schuurmans. Improved Natural Language Learning via Variance-Regularization Support Vector Machines. In *CoNLL*, 2010.

[27] R. Cuingnet, M. Chupin, H. Benali, and O. Colliot. Spatial and anatomical regularization of SVM for brain image analysis. In *NIPS*, 2010.

[28] P. Shivaswamy and T. Jebara. Maximum relative margin and data-dependent regularization. *JMLR*, 11:747–788, 2010.

[29] K. Gai, G. Chen, and C. Zhang. Learning kernels with radiuses of minimum enclosing balls. In *NIPS*, 2010.

[30] S. G. Mueller, M. W. Weiner, et al. Ways toward an early diagnosis in Alzheimers disease: The Alzheimer's Disease Neuroimaging Initiative. *J. of the Alzheimer's Association*, 1(1):55–66, 2005.

